# Neural Network On-Line Learning Control of Spacecraft Smart Structures

**Dr. Christopher Bowman**
Ball Aerospace Systems Group
P.O. Box 1062
Boulder, CO 80306

## Abstract

The overall goal is to reduce spacecraft weight, volume, and cost by on-line adaptive non-linear control of flexible structural components. The objective of this effort is to develop an adaptive Neural Network (NN) controller for the Ball C-Side 1m x 3m antenna with embedded actuators and the RAMS sensor system. A traditional optimal controller for the major modes is provided perturbations by the NN to compensate for unknown residual modes. On-line training of recurrent and feed-forward NN architectures have achieved adaptive vibration control with unknown modal variations and noisy measurements. On-line training feedback to each actuator NN output is computed via Newton's method to reduce the difference between desired and achieved antenna positions.

## 1 ADAPTIVE CONTROL BACKGROUND

The two traditional approaches to adaptive control are 1) direct control (such as performed in direct model reference adaptive controllers) and 2) indirect control (such as performed by explicit self-tuning regulators). Direct control techniques (e.g. model-reference adaptive control) provide good stability however are susceptible to noise. Whereas indirect control techniques (e.g. explicit self-tuning regulators) have low noise susceptibility and good convergence rate. However they require more control effort and have worse stability and are less robust to mismodeling. NNs synergistically augment traditional adaptive control techniques by providing improved mismodeling robustness both adaptively on-line for time-varying dynamics as well as in a learned control mode at a slower rate.

The NN control approaches which correspond to direct and indirect adaptive control are commonly known as inverse and forward modeling, respectively. More specifically, a NN which maps the plant state and its desired performance to the control command is called an inverse model, a NN mapping both the current plant state and control to the next state and its performance is called the forward model.

When given a desired performance and the current state, the inverse model generates the control, see Figure 1. The actual performance is observed and is used to train/update the inverse model. A significant problem occurs when the desired and achieved performance differ greatly since the model near the desired state is not changed. This condition is corrected by adding random noise to the control outputs so as to extend the state space

being explored. However, this correction has the effect of slowing the learning and reducing broadband stability.

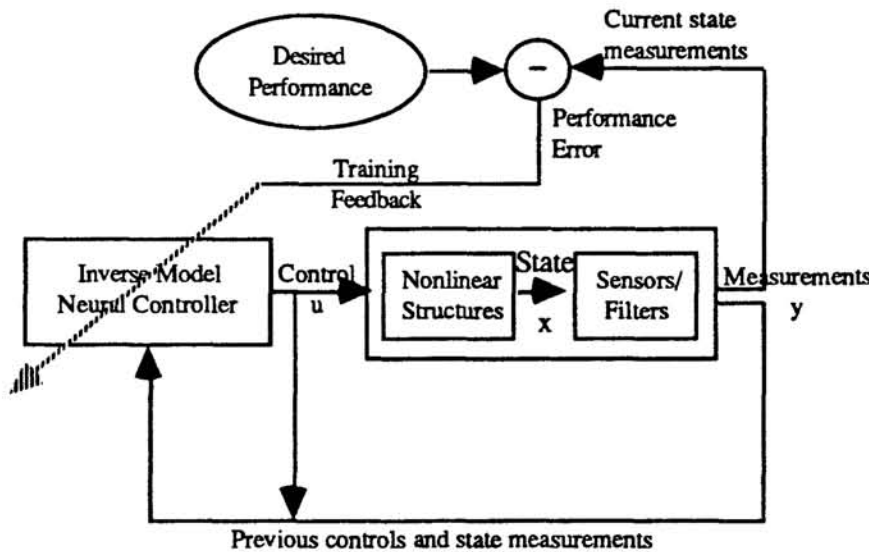

Figure 1: Direct Adaptive Control Using Inverse Modeling Neural Network Controller

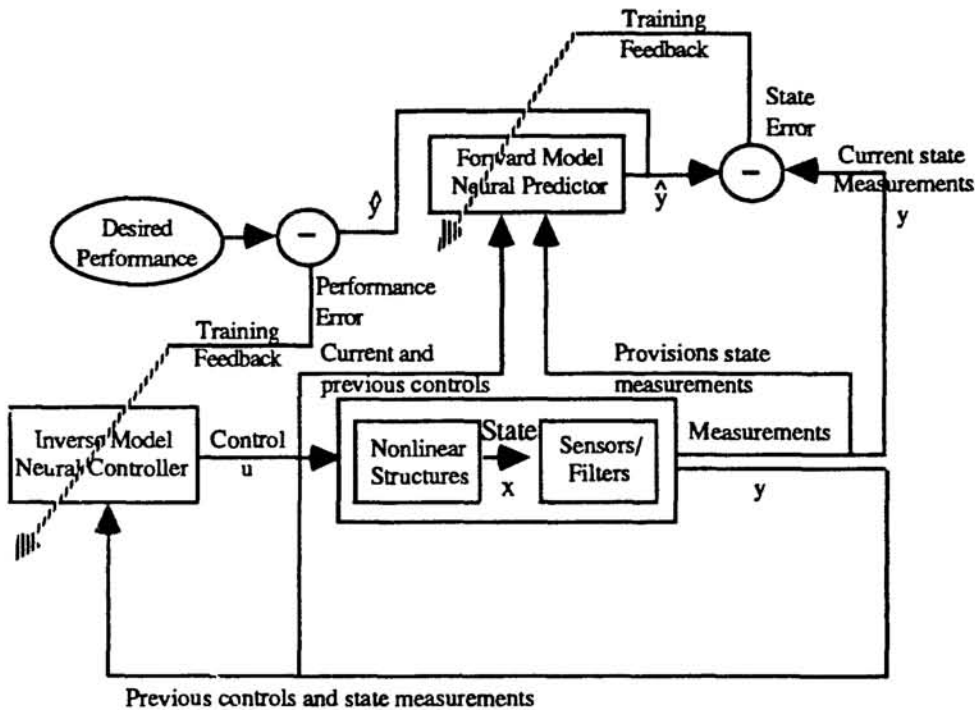

Figure 2:  Dual (Indirect and Direct) Adaptive Control Using Forward Modeling Neural Network State Predictor To Aid Inverse Model Convergence

For forward modeling the map from the current control and state to the resulting state and performance is learned, see Figure 2. For cases where the performance is evaluated at a future time (i.e. distal in time), a predictive critic [Barto and Sutton, 1989] NN model is learned. In both cases the Jacobian of this performance can be computed to iteratively generate the next control action. However, this differentiating of the critic NN for back-propagation training of the controller network is very slow and in some cases steers the searching the wrong direction due to initial erroneous forward model estimates. As the NN adapts itself the performance flattens which results in the slow halting of learning at an

unacceptable solution. Adding noise to the controller's output [Jordan and Jacobs, 1990] breaks the redundancy but forces the critic to predict the effects of future noise. This problem has been solved by using a separately trained intermediate plant model to predict the next state from the prior state and control while having an independent predictor model generate the performance evaluation from the plant model predicted state [Werbos, 1990] and [Brody, 1991]. The result is a 50-100 fold learning speed improvement over reinforcement training of the forward model controller NN.

However, this method still relies on a "good" forward model to incrementally train the inverse model. These incremental changes can still lead to undesirable solutions. For control systems which follow the stage 1, 2 or 3 models given in [Narendra, 1991) the control can be analytically computed from a forward-only model. For the most general, non-linear (stage 4) systems, an alternative is the memory-based forward model [Moore, 1992]. Using only a forward NN model, a direct hill-climbing or Newton's method search of candidate actions can be applied until a control decision is reached. The resulting state and its performance are used for on-line training of the forward model. Judicial random control actions are applied to improve behavior only where the forward model error is predicted to be large (e.g. via cross-validation). Also using robust regression, experiences can be deweighted according to their quality and their age. The high computational burden of these cross-validation techniques can be reduced by parallel on-line processing providing the "policy" parameters for fast on-line NN control.

For control problems which are distal in time and space, a hybrid of these two forward-modeling approaches can be used. Namely, a NN plant model is added which is trained off-line in real-time and updated as necessary at a slower rate than the on-line forward model which predicts performance based upon the current plant model. This slower rate trained forward-model NN supports learned control (e.g. via numerical inversion) whereas the on-line forward model provides the faster response adaptive control. Other NN control techniques such as using a Hopfield net to solve the optimal-control quadratic-programming problem or the supervised training of ART II off-line with adaptive vigilance for on-line pole placement have been proposed. However, their on-line robustness appears limited due to their sensitivity to a priori parameter assumptions.

A forward model NN which augments a traditional controller for unmodeled modes and unforeseen situations is presented in the following section. Performance results for both feed-forward and current learning versions are compared in Section 3.

## 2  RESIDUAL FORWARD MODEL NEURAL NETWORK (RFM-NN) CONTROLLER

A type of forward model NN which acts as a residual mode filter to support a reduced-order model (ROM) traditional optimal state controller has been evaluated, see Figure 3. The ROM determines the control based upon its model coordinate approximate representation of the structure. Model coordinates are obtained by a transformation using known primary vibration modes, [Young, 1990]. The transformation operator is a set of eigenvectors (mode shapes) generated by finite element modeling. The ROM controller is traditionally augmented by a residual-mode filter (RMF). Ball's RFM-NN replaces the RMF in order to better capture the mismodeled, unmodeled and changing modes.

The objective of the RFM-NN is to provide ROM controller with ROM derivative state perturbations, so that the ROM controls the structure as desired by the user. The RFM-NN is trained on-line using scored supervised feedback to generate these desired ROM state perturbations. The scored supervised training provides a score for each perturbation output based upon the measured position of the structure. The measured deviations, $Y^*(t)$, from the desired structure position are converted to errors in the estimated ROM state using the ROM transformation. Specifically, the training score, $S(t)$, for each ROM derivative state $\dot{x}_N(t)$ is expressed in the following discrete equation:

$$S(t) = B_N Y^*(t) - \hat{\dot{x}}_N(t)$$
$$\text{where } \hat{\dot{x}}_N(t) = \left[A_N + B_N G_N - K_N C_N\right]\hat{x}_N(t-1) + K_N Y(t-1)$$

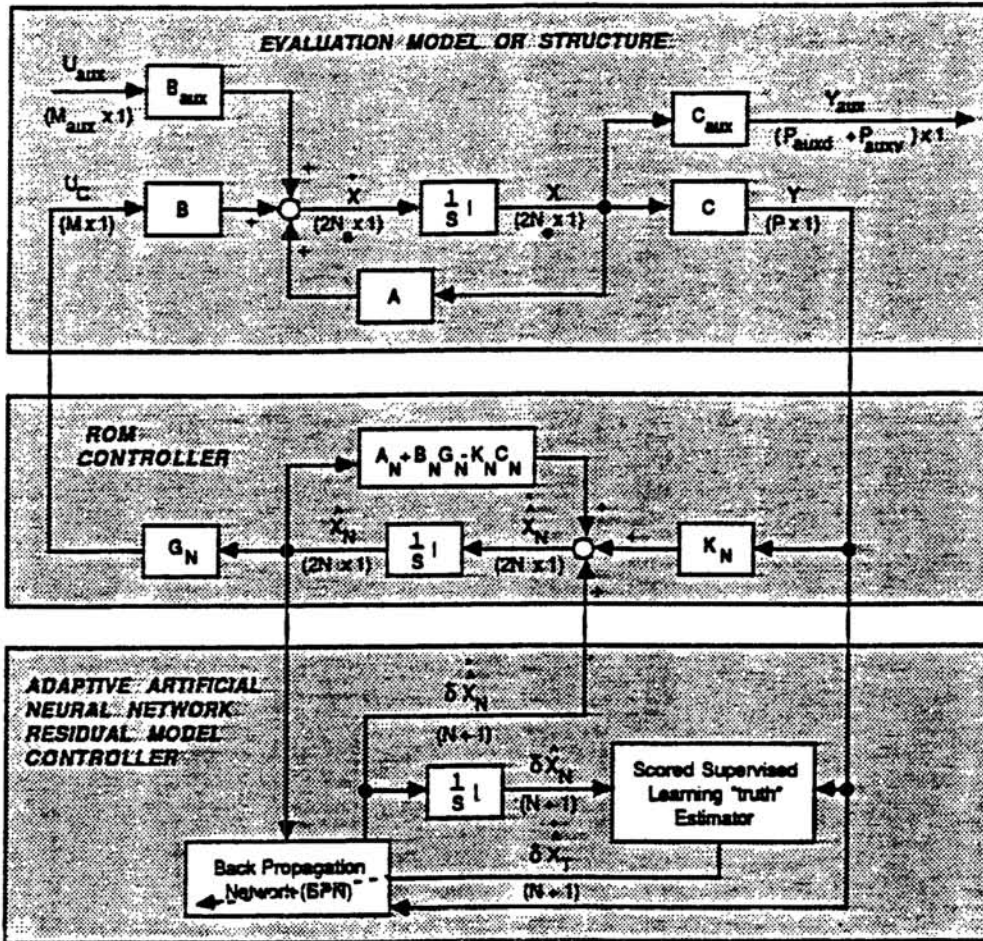

Figure 3:   Residual Forward Model Neural Network Adaptive Controller Replaces
Traditional Residual Mode Filter

Newton's method is then applied to find the $\hat{\delta x}_N(t)$ ROM state perterbations which zero
the score. First, the score is smoothed, $\bar{S}(t) = \delta \bar{S}(t-1) + (1-\delta)S(t)$ and the neural
network output is smoothed similarly. Second, Newton's method computes the
adjustments needed to zero the scores,

$$\Delta(\hat{\delta x}_N(t)) = -\bar{S}(t)(\hat{\delta x}_N(t) - \hat{\delta x}_N(t-1)) / [\bar{S}(t) - \bar{S}(t-1)]$$
$$= -\varepsilon x_N(t) \text{ (if either difference} = 0)$$

Third, the NN is trained, $\hat{\delta x}_T(t+1) = \alpha\Delta(\hat{\delta x}_N(t)) + \hat{\delta x}_N(t)$ with the appropriate
learning rate, $\alpha$ (e.g. approximation to inverse of largest eigenvalue of the Hessian
weight matrix).

## 3   RFM-NN ON-LINE LEARNING RESULTS

Both feed-forward and recurrent RFM-NNs have been incorporated into an interactive
simulation of Ball's Control-Structure Interaction Demonstration Experiment (C-SIDE)
see Figure 4. This 1m x 3m lightweight antenna facesheet has 8 embedded actuators plus
three auxiliary input actuators and uses 8 remote angular measurement sensors (RAMS)
plus 4 displacement and 3 velocity auxiliary sensors. In order to evaluate the on-line
performance of the RFM-NNs the ROM controller was given insufficient and partially
incorrect modes. The ROM without the RFM-NN grew unstable (i.e. greater than 10
millimeter C-SIDE displacements) in 13 seconds. The initial feed-forward RFM-NN used
8 sensor and 6 ROM state feedback estimate inputs as well as 5 hidden units and 3 ROM
velocity state perturbation outputs. This RFM-NN had random initial weights, logistic

activation functions, and back-propagation training using one sixth the learning rate for the output layers (e.g. .06 and .01). Newton RFM-NN training search used a step size of one with smoothing factor of one tenth.

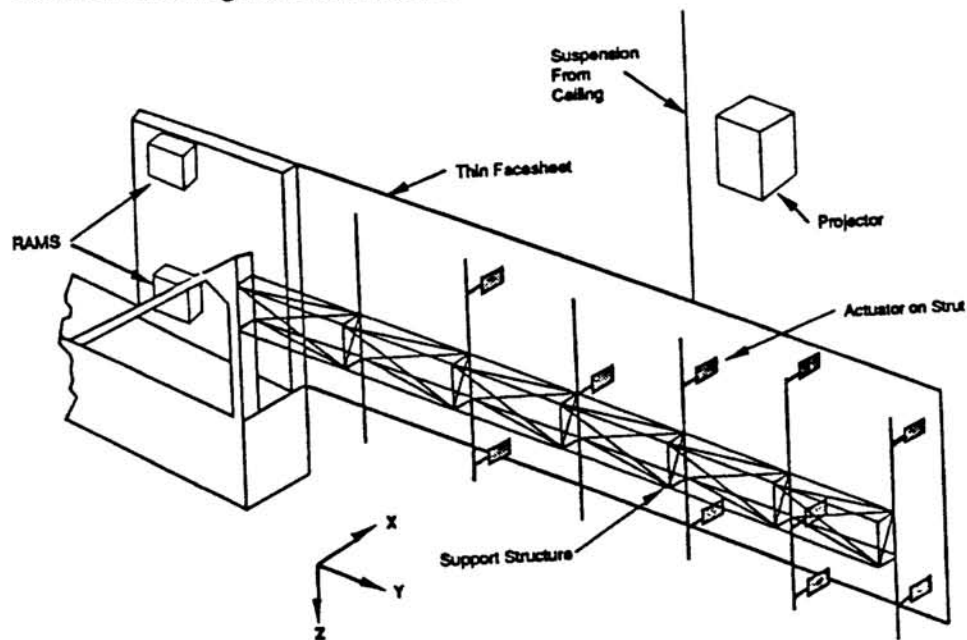

Figure 4: 1m x 3m C-SIDE Antenna Facesheet With Embedded Actuators.

This RFM-NN learned on-line to stabilized and reduce vibration to less than ±1mm within 20 seconds, see Figure 5. A five Newton force applied a few seconds later is compensated for within nine seconds, see Figure 6. This is accomplished with learning off as well as when on. To test the necessity of the RFM-NN the ROM was given the scored supervised training (i.e. Newton's search estimates) directly instead of the RFM-NN outputs. This caused immediate unstable behavior. To test the RFM-NN sensitivity to measurement accuracy a uniform error of ±5% was added. Starting from the same random weight start the RFM-NN required 25 seconds to learn to stabilize the antenna, see Figure 7. The best stability was achieved when the product of the Newton and BPN steps was approximately .01. This feed-forward NN was compared to an Elman-type recurrent NN (i.e. hidden layer feedback to itself with one-step BP training). The recurrent RFM-NN on-line learning stability was much less sensitive to initial weights. The recurrent RFM-NN stabilized C-SIDE with up to 10% - 20% measurement noise versus 5% - 10% limit for feed-forward RFM-NN.

## 4  SUMMARY AND RECOMMENDATIONS

Adaptive smart structures promise to reduce spacecraft weight and dependence on extensive ground monitoring. A recurrent forward model NN is used as a residual mode filter to augment a traditional reduced-order model (ROM) controller. It was more robust than the feed-forward NN and the traditional-only controller in the presence of unmodeled modes and noisy measurements. Further analyses and hardware implementations will be performed to better quantify this robustness including the sensitivity to the ROM controller mode fidelity, number of output modes, learning rates, measurement-to-state errors, and time quantization effects.

To improve robustness to ROM mode changes a comparison to the dual forward/inverse NN control approach is recommended. The forward model will adjust the search used to train an inverse model which provides control augmentations to the ROM controller. This will enable control searches to occur both off-line faster than real-time using the forward model (i.e. imagination) and on-line using direct search trials with varying noise levels. The forward model will adapt using quality experiences (e.g. via cross validation) which improves inverse models searches. The inverse model reliance on forward model will reduce until forward model prediction errors increase. Future challenges, include solving

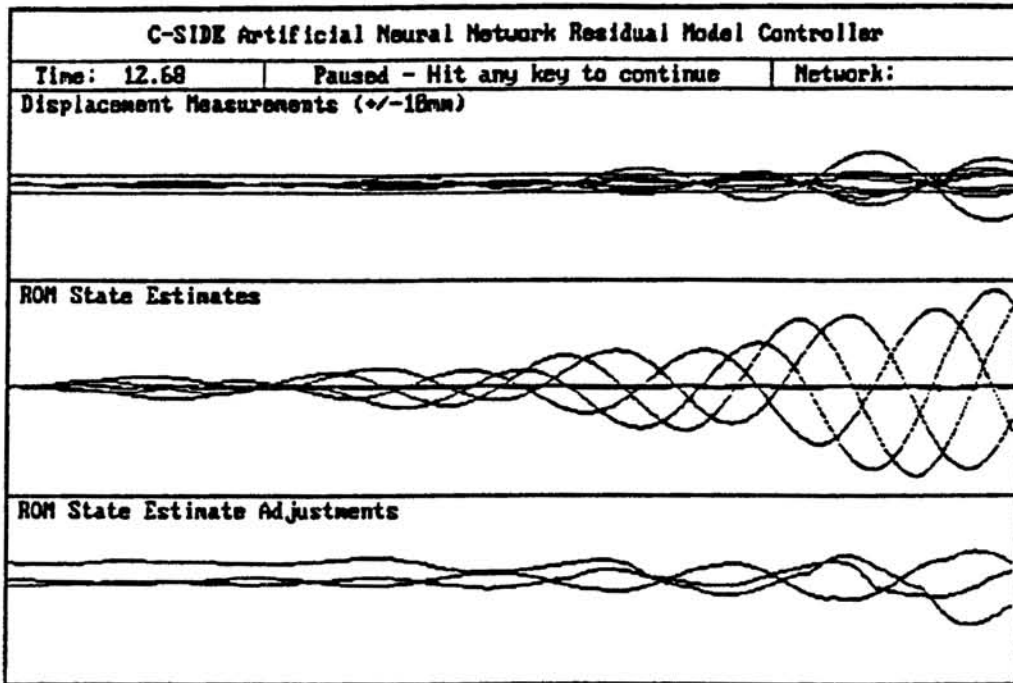

Figure 5: RFM-NN On-Line Learning To Achieve Stable Control

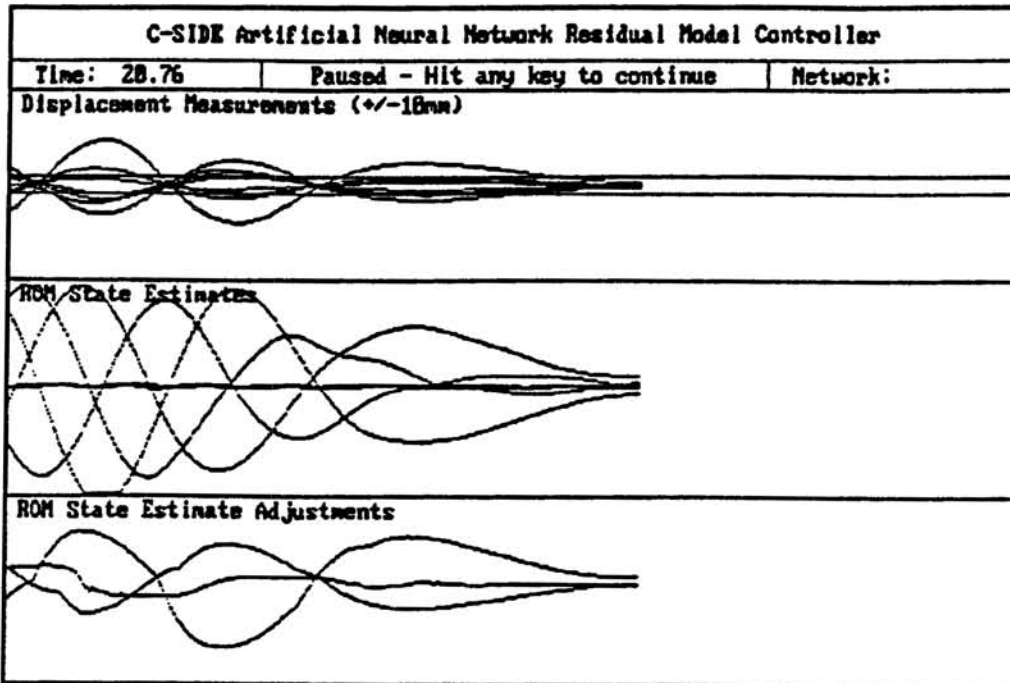

Figure 5: RFM-NN On-Line Learning To Achieve Stable Control (concluded)

the temporal credit assignment problem, partitioning to restricted chip sizes, combining with incomplete a priori knowledge, and balancing adaptivity of response with long-term learning. The goal is to extend stability-dominated, fixed-goal traditional control with adaptive robotic-type neural control to enable better autonomous control where fully-justified fixed models and complete system knowledge are not required. The resultant robust autonomous control will capitalize on the speed of massively parallel analog neural-like computations (e.g. with NN pulse stream chips).

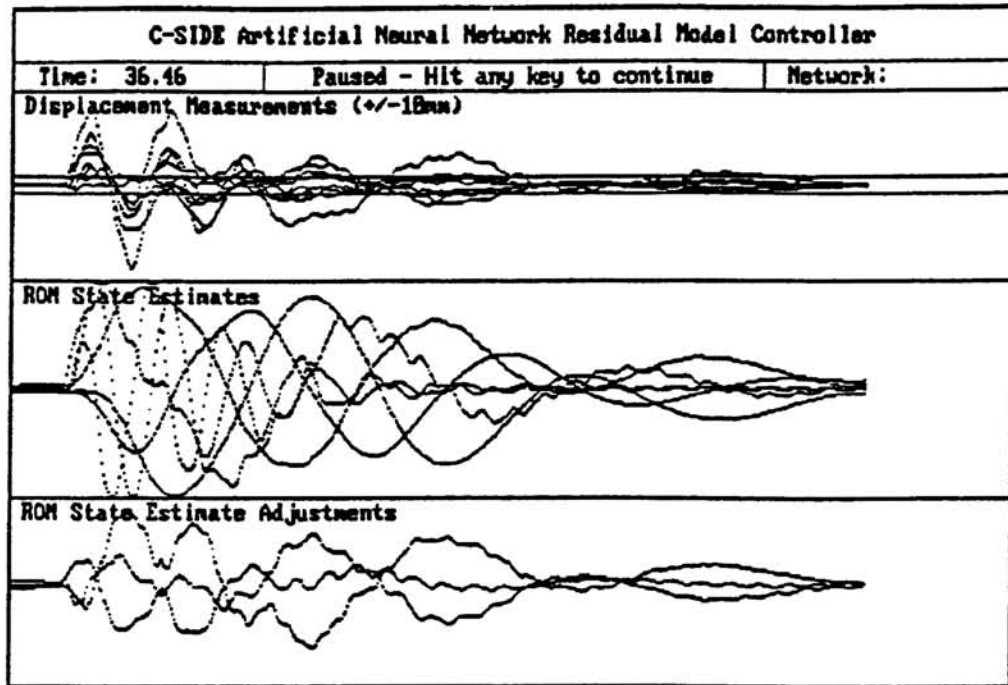

Figure 6:   5 Newton Force Vibration Removed Using RFM-NN Learned Forward Model

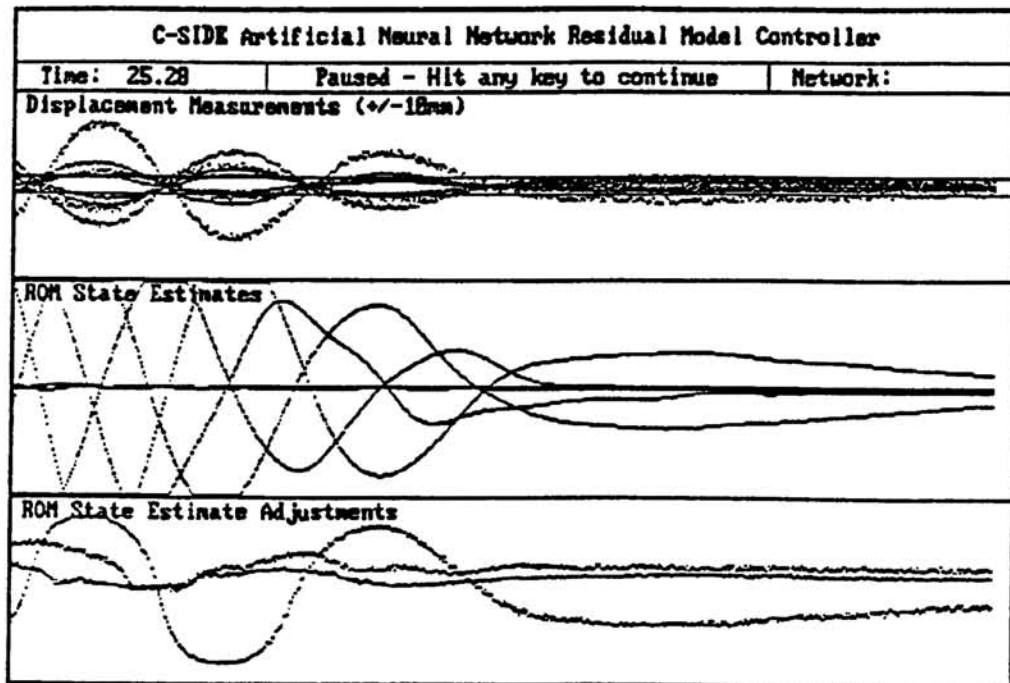

Figure 7:   RFM-NN Learning to Remove Vibrations in C-SIDE With ±15% Noisy
Displacement Measurements

## 5  REFERENCES

Barto, A.G., Sutton, R.S., and Watkins, C.J.C.H., *Learning and Sequential Decision Making*, Univ. of Mass. at Amherst COINS Technical Report 89-95, September 1989

Bowman, C.L., *Adaptive Neural Networks Applied to Signal Recognition*, 3rd Tri-Service Data fusion Symposium, May 1989

Brody, Carlos, *Fast Learning With Predictive Forward Models*, Neural Information Processing Systems 4 (NIPS4), 1992

Jorden, M.I., and Jacobs, R.A., *Learning to Control and Unstable System with Forward Modeling*, in D.S. Touretzky, ed., Advances in NIPS 2, Morgan Kaufmann 1990.

Moore, A.W., *Fast, Robust Adaptive Control by Learning Only Forward Models*, NIPS 4, 1992

Mukhopadhyay S. and Narendra, D.S., *Disturbance Rejection in Nonlinear Systems Using Neural Networks* Yale University Report No. 9114 December 1991

Werbos, P., *Architectures For Reinforcement Learning*, in Miller, Sutton and Werbos, ed., Neural Networks for Control, MIT Press 1990

Young, D.D., *Distributed Finite-Element Modeling and Control Approach for Large Flexible Structures*, J. of Guidance, Control and Dynamics, Vol. 13 (4), 703-713, 1990
